# From Algorithmic to Subjective Randomness

**Thomas L. Griffiths & Joshua B. Tenenbaum**
{gruffydd,jbt}@mit.edu
Massachusetts Institute of Technology
Cambridge, MA 02139

## Abstract

We explore the phenomena of subjective randomness as a case study in understanding how people discover structure embedded in noise. We present a rational account of randomness perception based on the statistical problem of model selection: given a stimulus, inferring whether the process that generated it was random or regular. Inspired by the mathematical definition of randomness given by Kolmogorov complexity, we characterize regularity in terms of a hierarchy of automata that augment a finite controller with different forms of memory. We find that the regularities detected in binary sequences depend upon presentation format, and that the kinds of automata that can identify these regularities are informative about the cognitive processes engaged by different formats.

## 1 Introduction

People are extremely good at finding structure embedded in noise. This sensitivity to patterns and regularities is at the heart of many of the inductive leaps characteristic of human cognition, such as identifying the words in a stream of sounds, or discovering the presence of a common cause underlying a set of events. These acts of everyday induction are quite different from the kind of inferences normally considered in machine learning and statistics: human cognition usually involves reaching strong conclusions on the basis of limited data, while many statistical analyses focus on the asymptotics of large samples.

The ability to detect structure embedded in noise has a paradoxical character: while it is an excellent example of the kind of inference at which people excel but machines fail, it also seems to be the source of errors in tasks at which machines regularly succeed. For example, a common demonstration conducted in introductory psychology classes involves presenting students with two binary sequences of the same length, such as HHTHTHTT and HHHHHHHH, and asking them to judge which one seems more random. When students select the former, they are told that their judgments are irrational: the two sequences are equally random, since they have the same probability of being produced by a fair coin. In the real world, the sense that some random sequences seem more structured than others can lead people to a variety of erroneous inferences, whether in a casino or thinking about patterns of births and deaths in a hospital [1].

Here we show how this paradox can be resolved through a proper understanding of what our sense of randomness is designed to compute. We will argue that our sense of randomness is actually extremely well-calibrated with a rational statistical computation – just not the one to which it is usually compared. While previous accounts criticize people's randomness

judgments as poor estimates of the probability of an outcome, we claim that subjective randomness, together with other everyday inductive leaps, can be understood in terms of the statistical problem of *model selection*: given a set of data, evaluating hypotheses about the process that generated it. Solving this model selection problem for small datasets requires two ingredients: a set of hypotheses about the processes by which the data could have been generated, and a rational statistical inference by which these hypotheses are evaluated.

We will model subjective randomness as an inference comparing the probability of a sequence under a random process, $P(X|\text{random})$, with the probability of that sequence under a regular process, $P(X|\text{regular})$. In previous work we have shown that defining $P(X|\text{regular})$ using a restricted form of Kolmogorov complexity, in which regularity is characterized in terms of a simple computing machine, can provide a good account of human randomness judgments for binary sequences [2]. Here, we explore the consequences of manipulating the conditions under which these sequences are presented. We will show that the kinds of regularity to which people are sensitive depend upon whether the full sequence is presented simultaneously, or its elements are presented sequentially. By exploring how these regularities can be captured by different kinds of automata, we extend our rational analysis of the *inference* involved in subjective randomness to a rational characterization of the *processes* underlying it: certain regularities can only be detected by automata with a particular form of memory access, and identifying the conditions under which regularities are detectable provides insight into how characteristics of human memory interact with rational statistical inference.

## 2 Kolmogorov complexity and randomness

A natural starting point for a formal account of subjective randomness is Kolmogorov complexity, which provides a mathematical definition of the randomness of a sequence in terms of the length of the shortest computer program that would produce that sequence. The idea of using a code based upon the length of computer programs was independently proposed in [3], [4] and [5], although it has come to be associated with Kolmogorov. A sequence $X$ has Kolmogorov complexity $K(X)$ equal to the length of the shortest program $p$ for a (prefix) universal Turing machine $U$ that produces $X$ and then halts,

$$K(X) = \min_{p:U(p)=X} \ell(p), \tag{1}$$

where $\ell(p)$ is the length of $p$ in bits. Kolmogorov complexity identifies a sequence $X$ as random if $\ell(X) - K(X)$ is small: random sequences are those that are irreducibly complex [4]. While not necessarily following the form of this definition, psychologists have preserved its spirit in proposing that the perceived randomness of a sequence increases with its complexity (eg. [6]). Kolmogorov complexity can also be used to define a variety of probability distributions, assigning probability to events based upon their complexity. One such distribution is *algorithmic probability*, in which the probability of $X$ is

$$R(X) = 2^{-K(X)} = \max_{p:U(p)=X} 2^{-\ell(p)}. \tag{2}$$

There is no requirement that $R(X)$ sum to one over all sequences; many probability distributions that correspond to codes are unnormalized, assigning the missing probability to an undefined sequence.

There are three problems with using Kolmogorov complexity as the basis for a computational model of subjective randomness. Firstly, the Kolmogorov complexity of any particular sequence $X$ is not computable [4], presenting a practical challenge for any modelling effort. Secondly, while the universality of an encoding scheme based on Turing machines is attractive, many of the interesting questions in cognition come from the details: issues of representation and processing are lost in the asymptotic equivalence of coding schemes, but

play a key role in people's judgments. Finally, Kolmogorov complexity is too permissive in what it considers a regularity. The set of regularities identified by people are a strict subset of those that might be expressed in short computer programs. For example, people are very unlikely to be able to tell the difference between a binary sequence produced by a linear congruential random number generator (a very short program) and a sequence produced by flipping a coin, but these sequences should differ significantly in Kolmogorov complexity. Restricting the set of regularities does not imply that people are worse than machines at recognizing patterns: reducing the size of the set of hypotheses increases inductive bias, making it possible to identify the presence of structure from smaller samples.

## 3 A statistical account of subjective randomness

While there are problems with using Kolmogorov complexity as the basis for a rational theory of subjective randomness, it provides a clear definition of regularity. In this section we will present a statistical account of subjective randomness in terms of a comparison between random and regular sources, where regularity is defined by analogues of Kolmogorov complexity for simpler computing machines.

### 3.1 Subjective randomness as model selection

One of the most basic problems that arises in statistical inference is identifying the source of a set of observations, based upon a set of hypotheses. This is the problem of model selection. Model selection provides a natural basis for a statistical theory of subjective randomness, viewing these judgments as the consequence of an inference to the process that produced a set of observations. On seeing a stimulus $X$, we consider two hypotheses: $X$ was produced by a random process, or $X$ was produced by a regular process. The decision about the source of $X$ can be formalized as a Bayesian inference,

$$\frac{P(\text{random}|X)}{P(\text{regular}|X)} = \frac{P(X|\text{random})}{P(X|\text{regular})} \frac{P(\text{random})}{P(\text{regular})}, \tag{3}$$

in which the posterior odds in favor of a random generating process are obtained from the likelihood ratio and the prior odds. The only part of the right hand side of the equation affected by $X$ is the likelihood ratio, so we define the subjective randomness of $X$ as

$$\text{random}(X) = \log \frac{P(X|\text{random})}{P(X|\text{regular})}, \tag{4}$$

being the evidence that $X$ provides towards the conclusion that it was produced by a random process.

### 3.2 The nature of regularity

In order to define $\text{random}(X)$, we need to specify $P(X|\text{random})$ and $P(X|\text{regular})$. When evaluating binary sequences, it is natural to set $P(X|\text{random}) = (\frac{1}{2})^{\ell(X)}$. Taking the logarithm in base 2, $\text{random}(X)$ is $-\ell(X) - \log_2 P(X|\text{regular})$, depending entirely on $P(X|\text{regular})$. We obtain $\text{random}(X) = K(X) - \ell(X)$, the difference between the complexity of a sequence and its length, if we choose $P(X|\text{regular}) = R(X)$, the algorithmic probability defined in Equation 2. This is identical to the mathematical definition of randomness given by Kolmogorov complexity. However, the key point of this statistical approach is that we are not restricted to using $R(X)$: we have a measure of the randomness of $X$ for any choice of $P(X|\text{regular})$.

The choice of $P(X|\text{regular})$ will reflect the stimulus domain, and express the kinds of regularity which people can detect in that domain. For binary sequences, a good candidate for specifying $P(X|\text{regular})$ is a hidden Markov model (HMM), a probabilistic finite

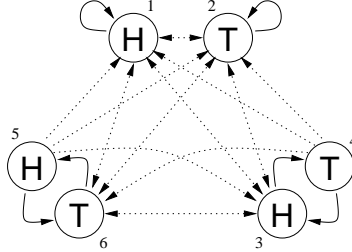

Figure 1: Finite state automaton used to define $P(X|\text{regular})$ to give $\text{random}(X) \propto DP$. Solid arrows are transitions consistent with repeating a motif, which are taken with probability $\delta$. Dashed arrows are motif changes, using the prior determined by $\alpha$.

state automaton. In fact, specifying $P(X|\text{regular})$ in terms of a particular HMM results in $\text{random}(X)$ being equivalent to the "Difficulty Predictor" (DP) [6] a measure of sequence complexity that has been extremely successful in modelling subjective randomness judgments. DP measures the complexity of a sequence in terms of the number of repeating (eg. HHHH) and alternating (eg. HTHT) subsequences it contains, adding one point for each repeating subsequence and two points for each alternating subsequence. For example, the sequence TTTHHHTHTH is a run of tails, a run of heads, and an alternating sub-sequence, $DP = 4$. If there are several partitions into runs and alternations, $DP$ is calculated on the partition that results in the lowest score.

In [2], we showed that $\text{random}(X) \propto DP$ if $P(X|\text{regular})$ is specified by a particular HMM. This HMM produces sequences by motif repetition, using the transition graph shown in Figure 1. The model emits sequences by choosing a motif, a sequence of symbols of length $k$, with probability proportional to $\alpha^k$, and emitting symbols consistent with that motif with probability $\delta$, switching to a new motif with probability $1 - \delta$. In Figure 1, state 1 repeats the motif H, state 2 repeats T, and the remaining states repeat the alternating motifs HT and TH. The randomness of a sequence under this definition of regularity depends on $\delta$ and $\alpha$, but is generally affected by the number of repeating and alternating subsequences. The equivalence to DP, in which a sequence scores a single point for each repeating subsequence and two points for each alternating subsequence, results from taking $\delta = 0.5$ and $\alpha = \frac{\sqrt{3}-1}{2}$, and choosing the the state sequence for the HMM that maximizes the probability of the sequence.

Just as the algorithmic probability $R(X)$ is a probability distribution defined by the length of programs for a universal Turing machine, this choice of $P(X|\text{regular})$ can be seen as specifying the length of "programs" for a particular finite state automaton. The output of a finite state automaton is determined by its state sequence, just as the output of a universal Turing machine is determined by its program. However, since the state sequence is the same length as the sequence itself, this alone does not provide a meaningful measure of complexity. In our model, probability imposes a metric on state sequences, dictating a greater cost for moves between certain states, which translates into a code length through the logarithm. Since we find the state sequence most likely to have produced $X$, and thus the shortest code length, we have an analogue of Kolmogorov complexity defined on a finite state automaton.

### 3.3 Regularities and automata

Using a hidden Markov model to specify $P(X|\text{regular})$ provides a measure of complexity defined in terms of a finite state automaton. However, the kinds of regularities people can detect in binary sequences go beyond the capacity of a finite state automaton. Here, we consider three additional regularities: symmetry (eg. THTHHHTHT), symmetry in the com-

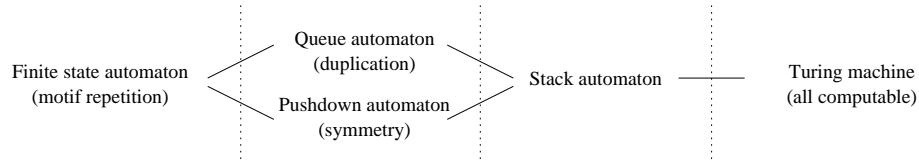

Figure 2: Hierarchy of automata used to define measures of complexity. Of the regularities discussed in this paper, each automaton can identify all regularities identified by those automata to its left as well as those stated in parentheses beneath its name.

plement (eg. TTTTHHHH), and the perfect duplication of subsequences (eg. HHHTHHHT vs. HHHTHHHTH). These regularities identify formal languages that cannot be recognized by a finite state automaton, suggesting that we might be able to develop better models of subjective randomness by defining $P(X|\text{regular})$ in terms of more sophisticated automata.

The automata we will consider in this paper form a hierarchy, shown in Figure 2. This hierarchy expresses the same content as Chomsky's [7] hierarchy of computing machines – the regularities identifiable by each machine are a strict superset of those identifiable to the machine to the left – although it features a different set of automata. The most restricted set of regularities are those associated with the finite state automaton, and the least restricted are those associated with the Turing machine. In between are the pushdown automaton, which augments a finite controller with a stack memory, in which the last item added is the first to be accessed; the queue automaton,[1] in which the memory is a queue, in which the first item added is the first to be accessed; and the stack automaton, in which the memory is a stack but any item in the stack can be read by the controller [9, 10]. The key difference between these kinds of automata is the memory available to the finite controller, and exploring measures of complexity defined in terms of these automata thus involves assessing the kind of memory required to identify regularities.

Each of the automata shown in Figure 2 can identify a different set of regularities. The finite state automaton is only capable of identifying motif repetition, while the pushdown automaton can identify both kinds of symmetry, and the queue automaton can identify duplication. The stack automaton can identify all of these regularities, and the Turing machine can identify all computable regularities. For each of the sub-Turing automata, we can use these constraints to specify a probabilistic model for $P(X|\text{regular})$. For example, the probabilistic model corresponding to the pushdown automaton generates regular sequences by three methods: repetition, producing sequences with probabilities determined by the HMM introduced above; symmetry, where half of the sequence is produced by the HMM and the second half is produced by reflection; and complement symmetry, where the second half is produced by reflection and exchanging H and T. We then take $P(X|\text{regular}) = \max_{Z,M} P(X, Z|M)P(M)$, where $M$ is the method of production and $Z$ is the state sequence for the HMM. Similar models can be defined for the queue and stack automata, with the queue automaton allowing generation by repetition or duplication, and the stack automaton allowing any of these four methods. Each regularity introduced into the model requires a further parameter in specifying $P(M)$, so the hierarchy shown in Figure 2 also expresses the statistical structure of this set of models: each model is a special case of the model to its right, in which some regularities are eliminated by setting $P(M)$ to zero. We can use this structure to perform model selection with likelihood ratio tests, determining which model gives the best account of a particular dataset using just the difference in the log-likelihoods. We apply this method in the next section.

# 4    Testing the models

The models introduced in the previous section differ in the memory systems with which they augment the finite controller. The appropriateness of any one measure of complexity to a particular task may thus depend upon the memory demands placed upon the participant. To explore this hypothesis, we conducted an experiment in which participants make randomness judgments after either seeing a sequence in its entirety, or seeing each element one after another. We then used model selection to determine which measure of complexity gave the best account of each condition, illustrating how the strategy of defining more restricted forms of complexity can shed light into the cognitive processes underlying regularity detection.

## 4.1    Experimental methods

There were two conditions in the experiment, corresponding to Simultaneous and Sequential presentation of stimuli. The stimuli were sequences of heads (H) and tails (T) presented in 130 point fixed width sans-serif font on a 19" monitor at $1280 \times 1024$ pixel resolution. In the Simultaneous condition, all eight elements of the sequence appeared on the display simultaneously. In the Sequential condition, the elements appeared one by one, being displayed for 300ms with a 300ms inter-stimulus interval.

The participants were 40 MIT undergraduates, randomly assigned to the two conditions. Participants were instructed that they were about to see sequences which had either been produced by a random process (flipping a fair coin) or by other processes in which the choice of heads and tails was not random, and had to classify these sequences according to their source. After a practice session, each participant classified all 128 sequences of length 8, in random order, with each sequence randomly starting with either a head or a tail. Participants took breaks at intervals of 32 sequences.

## 4.2    Results and Discussion

We analyzed the results by fitting the models corresponding to the four automata described above, using all motifs up to length 4 to specify the basic model. We computed $\mathrm{random}(X)$ for each stimulus as in Eq. (4), with $P(X|\mathrm{regular})$ specified by the probabilistic model corresponding to each of the automata. We then converted this log-likelihood ratio into the posterior probability of a random generating process, using

$$P(\mathrm{random}|X) = \frac{1}{1 + \exp\{-\lambda\,\mathrm{random}(X) - \psi\}}$$

where $\lambda$ and $\psi$ are parameters weighting the contribution of the likelihoods and the priors respectively. We then optimized $\lambda, \psi, \delta, \alpha$ and the parameters contributing to $P(M)$ for each model, maximizing the likelihood of the classifications of the sequences by the 20 participants in each of the 2 conditions. The results of the model-fitting are shown in Figure 3(a) and (b), which indicate the relationship between the posterior probabilities predicted by the model and the proportion of participants who classified a sequence as random. The correlation coefficients shown in the figure provide a relatively good indicator of the fit of the models, and each sequence is labelled according to the regularity it expresses, showing how accommodating particular regularities contributes to the fit.

The log-likelihood scores obtained from fitting the models can be used for model selection, testing whether any of the parameters involved in the models are unnecessary. Since the models form a nested hierarchy, we can use likelihood ratio tests to evaluate whether introducing a particular regularity (and the parameters associated with it) results in a statistically significant improvement in fit. Specifically, if model 1 has log-likelihood $L_1$ and $df_1$ parameters, and model 2 has log-likelihood $L_2$ and $df_2 > df_1$ parameters, $2(L_2 - L_1)$

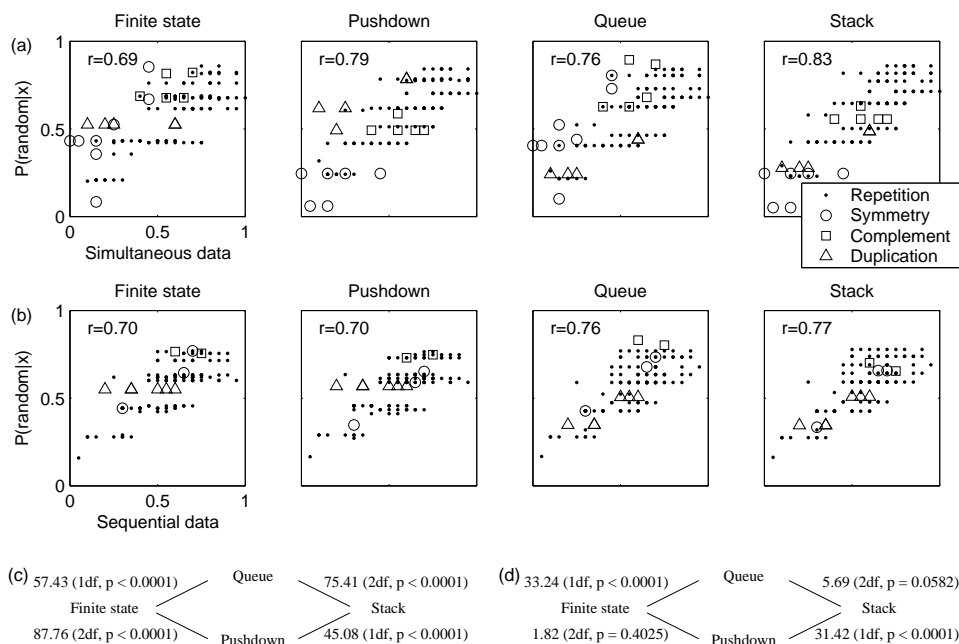

Figure 3: Experimental results for (a) the Simultaneous and (b) the Sequential condition, showing the proportion of participants classifying a sequence as "random" (horizontal axis) and $P(\text{random}|X)$ (vertical axis) as assessed by the four models. Points are labelled according to their parse under the Stack model. (c) and (d) show the model selection results for the Simultaneous and Sequential conditions respectively, showing the four automata with edges between them labelled with $\chi^2$ score (df, p-value) for improvement in fit.

should have a $\chi^2(df_2 - df_1)$ distribution under the null hypothesis of no improvement in fit. We evaluated the pairwise likelihood ratio tests for the four models in each condition, with the results shown in Figure 3(c) and (d). Additional regularities always improved the fit for the Simultaneous condition, while adding duplication, but not symmetry, resulted in a statistically significant improvement in the Sequential condition.

The model selection results suggest that the best model for the Simultaneous condition is the stack automaton, while the best model for the Sequential condition is the queue automaton. These results indicate the importance of presentation format in determining subjective randomness, as well as the benefits of exploring measures of complexity defined in terms of a range of computing machines. The stack automaton can evaluate regularities that require checking information in arbitrary positions in a sequence, something that is facilitated by a display in which the entire sequence is available. In contrast, the queue automaton can only access information in the order that it enters memory, and gives a better match to the task in which working memory is required. This illustrates an important fact about cognition – that human working memory operates like a queue rather than a stack – that is highlighted by this approach.

The final parameters of the best-fitting models provide some insight into the relative importance of the different kinds of regularities under different presentation conditions. For the Simultaneous condition, $\delta = 0.66, \alpha = 0.12, \lambda = 0.26, \psi = -1.98$ and motif repetition, symmetry, symmetry in the complement, and duplication were given probabilities of 0.748, 0.208, 0.005, and 0.039 respectively. Symmetry is thus a far stronger characteristic of reg-

ularity than either symmetry in the complement or duplication, when entire sequences are viewed simultaneously. For the Sequential condition, $\delta = 0.70, \alpha = 0.11, \lambda = 0.38, \psi = -1.24$, and motif repetition was given a probability of $0.962$ while duplication had a probability of $0.038$, with both forms of symmetry being given zero probability since the queue model provided the best fit. Values of $\delta > 0.5$ for both models indicates that regular sequences tend to repeat motifs, rather than rapidly switching between them, and the low $\alpha$ values reflect a preference for short motifs.

## 5   Conclusion

We have outlined a framework for understanding the rational basis of the human ability to find structure embedded in noise, viewing this inference in terms of the statistical problem of model selection. Solving this problem for small datasets requires two ingredients: strong prior beliefs about the hypothetical mechanisms by which the data could have been generated, and a rational statistical inference by which these hypotheses are evaluated. When assessing the randomness of binary sequences, which involves comparing random and regular sources, people's beliefs about the nature of regularity can be expressed in terms of probabilistic versions of simple computing machines. Different machines capture regularity when sequences are presented simultaneously and when their elements are presented sequentially, and the differences between these machines provide insight into the cognitive processes involved in the task. Analyses of the rational basis of human inference typically either ignore questions about processing or introduce them as relatively arbitrary constraints. Here, we are able to give a rational characterization of process as well as inference, evaluating a set of alternatives that all correspond to restrictions of Kolmogorov complexity to simple general-purpose automata.

**Acknowledgments.** This work was supported by a Stanford Graduate Fellowship to the first author. We thank Charles Kemp and Michael Lee for useful comments.

## Footnotes

[1]An unrestricted queue automaton is equivalent to a Turing machine. We will use the phrase to refer to an automaton in which the number of queue operations that can be performed for each input symbol is limited, which is generally termed a quasi real time queue automaton [8].

## References

[1] D. Kahneman and A. Tversky. Subjective probability: A judgment of representativeness. *Cognitive Psychology*, 3:430–454, 1972.

[2] T. L. Griffiths and J. B. Tenenbaum. Probability, algorithmic complexity and subjective randomness. In *Proceedings of the 25th Annual Conference of the Cognitive Science Society*, Hillsdale, NJ, 2003. Erlbaum.

[3] R. J. Solomonoff. A formal theory of inductive inference. Part I. *Information and Control*, 7:1–22, 1964.

[4] A. N. Kolmogorov. Three approaches to the quantitative definition of information. *Problems of Information Transmission*, 1:1–7, 1965.

[5] G. J. Chaitin. On the length of programs for computing finite binary sequences: statistical considerations. *Journal of the ACM*, 16:145–159, 1969.

[6] R. Falk and C. Konold. Making sense of randomness: Implicit encoding as a bias for judgment. *Psychological Review*, 104:301–318, 1997.

[7] N. Chomsky. Threee models for the description of language. *IRE Transactions on Information Theory*, 2:113–124, 1956.

[8] A. Cherubini, C. Citrini, S. C. Reghizzi, and D. Mandrioli. QRT FIFO automata, breadth-first grammars and their relations. *Theoretical Comptuer Science*, 85:171–203, 1991.

[9] S. Ginsburg, S. A. Greibach, and M. A. Harrison. Stack automata and compiling. *Journal of the ACM*, 14:172–201, 1967.

[10] A. V. Aho. Indexed grammars – an extension of context-free grammars. *Journal of the ACM*, 15:647–671, 1968.
